# Integrate-and-Fire models with adaptation are good enough: predicting spike times under random current injection

**Renaud Jolivet**\*
Brain Mind Institute, EPFL
CH-1015 Lausanne, Switzerland
*renaud.jolivet@epfl.ch*

**Alexander Rauch**
MPI for Biological Cybernetics
D-72012 Tübingen, Germany
*alexander.rauch@tuebingen.mpg.de*

**Hans-Rudolf Lüscher**
Institute of Physiology
CH-3012 Bern, Switzerland
*luescher@pyl.unibe.ch*

**Wulfram Gerstner**
Brain Mind Institute, EPFL
CH-1015 Lausanne, Switzerland
*wulfram.gerstner@epfl.ch*

## Abstract

Integrate-and-Fire-type models are usually criticized because of their simplicity. On the other hand, the Integrate-and-Fire model is the basis of most of the theoretical studies on spiking neuron models. Here, we develop a sequential procedure to quantitatively evaluate an equivalent Integrate-and-Fire-type model based on intracellular recordings of cortical pyramidal neurons. We find that the resulting effective model is sufficient to predict the spike train of the real pyramidal neuron with high accuracy. In in vivo-like regimes, predicted and recorded traces are almost indistinguishable and a significant part of the spikes can be predicted at the correct timing. Slow processes like spike-frequency adaptation are shown to be a key feature in this context since they are necessary for the model to connect between different driving regimes.

## 1 Introduction

In a recent paper, Feng [1] was questioning the "goodness" of the Integrate-and-Fire model (I&F). This is a question of importance since the I&F model is one of the most commonly used spiking neuron model in theoretical studies as well as in the machine learning community (see [2-3] for a review). The I&F model is usually criticized in the biological community because of its simplicity. It is believed to be much too simple to capture the firing dynamics of real neurons beyond a very rough and conceptual description of input integration and spikes initiation.

Nevertheless, recent years have seen several groups reporting that this type of model yields quantitative predictions of the activity of real neurons. Rauch and colleagues have shown that I&F-type models (with adaptation) reliably predict the mean firing rate of cortical

pyramidal cells [4]. Keat and colleagues have shown that a similar model is able to predict almost exactly the timing of spikes of neurons in the visual pathway [5]. However, the question is still open of how the predictions of I&F-type models compare to the precise structure of spike trains in the cortex. Indeed, cortical pyramidal neurons are known to produce spike trains whose reliability highly depends on the input scenario [6].

The aim of this paper is twofold. Firstly, we will show that there exists a systematic way to extract relevant parameters of an I&F-type model from intracellular recordings. To do so, we will follow the method exposed in [7] and which is based on optimal filtering techniques. Alternative approaches like maximum-likelihood methods exist and have been explored recently by Paninski and colleagues [8]. Note that both approaches had already been mentioned by Brillinger and Segundo [9]. Secondly, we will show by a quantitative evaluation of the model performances that the quality of simple threshold models is surprisingly good and is close to the intrinsic reliability of real neurons. We will try to convince the reader that, given the addition of a slow process, the I&F model is in fact a model that can be considered good enough for pyramidal neurons of the neocortex under random current injection.

## 2 Model and Methods

We started by collecting recordings. Layer 5 pyramidal neurons of the rat neocortex were recorded intracellularly in vitro while stimulated at the soma by a randomly fluctuating current generated by an Ornstein-Uhlenbeck (OU) process with a $1\,\mathrm{ms}$ autocorrelation time. Both the mean $\mu_I$ and the variance $\sigma_I^2$ of the OU process were varied in order to sample the response of the neurons to various levels of tonic and noisy inputs. Details of the experimental procedure can be found in [4]. A subset of these recordings was used to construct, separately for each recorded neuron, a generalized I&F-type model that we formulated in the framework of the Spike Response Model [3].

### 2.1 Definition of the model

The Spike Response Model (SRM) is written

$$u(t) = \eta(t - \hat{t}) + \int_0^{+\infty} \kappa(s)\, I(t - s)\mathrm{d}s \tag{1}$$

with $u$ the membrane voltage of the neuron and $I$ the external driving current. The kernel $\kappa$ models the integrative properties of the membrane. The kernel $\eta$ acts as a template for the shape of spikes (usually highly stereotyped). Like in the I&F model, the model neuron fires each time that the membrane voltage $u$ crosses the threshold $\vartheta$ from below

$$\text{if } u(t) \geq \vartheta(t) \text{ and } \frac{d}{dt}u(t) \geq \frac{d}{dt}\vartheta(t), \text{ then } \hat{t} = t \tag{2}$$

Here, the threshold includes a mechanism of spike-frequency adaptation. $\vartheta$ is given by the following equation

$$\frac{\mathrm{d}\vartheta}{\mathrm{d}t} = -\frac{\vartheta - \vartheta_0}{\tau_\vartheta} + A_\vartheta \sum_k \delta(t - t_k) \tag{3}$$

Each time that a spike is fired, the threshold $\vartheta$ is increased by a fixed amount $A_\vartheta$. It then decays back to its resting value $\vartheta_0$ with time constant $\tau_\vartheta$. $t_k$ denote the past firing times of the model neuron. During discharge at rate $f$, the threshold fluctuates around the average value

$$\bar{\vartheta} \approx \vartheta_0 + \alpha\, f \tag{4}$$

where $\alpha = A_\vartheta \, \tau_\vartheta$. This type of adaptation mechanism has been shown to constitute a universal model for spike-frequency adaptation [10] and has already been applied in a similar context [11]. During the model estimation, we use as a first step a traditional constant threshold denoted by $\vartheta(t) = \vartheta_{\text{cst}}$ which is then transformed in the adaptive threshold of Equation (3) by a procedure to be detailed below.

## 2.2 Mapping technique

The mapping technique itself is extensively described in [7,12-13] and we refer interested readers to these publications. In short, it is a systematic step-by-step evaluation and optimization procedure based on intracellular recordings. It consists in sequentially evaluating kernels ($\eta$ and $\kappa$) and parameters [$A_\vartheta$, $\vartheta_0$ and $\tau_\vartheta$ in Equation (3)] that characterize a specific instance of the model. The consecutive steps of the procedure are as follows

1. Extract the kernel $\eta$ from a sample voltage recording by spike triggered averaging. For the sake of simplicity, we assume that the mean drive $\mu_I = 0$.

2. Subtract $\eta$ from the voltage recording to isolate the subthreshold fluctuations.

3. Extract the kernel $\kappa$ by the Wiener-Hopf optimal filtering technique [7,14]. This step involves a comparison between the subthreshold fluctuations and the corresponding input current.

4. Find the optimal constant threshold $\vartheta_{\text{cst}}$. The optimal value of $\vartheta_{\text{cst}}$ is the one that maximizes the coefficient $\Gamma$ (see subsection 2.3 below for the definition of $\Gamma$). The parameter $\vartheta_{\text{cst}}$ depends on the specific set of input parameters (mean $\mu_I$ and variance $\sigma_I^2$) used during stimulation.

5. Plot the threshold $\vartheta_{\text{cst}}$ as a function of the firing frequency $f$ of the neuron and run a linear regression. $\vartheta_0$ is identified with the value of the fit at $f = 0$ and $\alpha$ with the slope [see Equation (4) and Figure 1C].

6. Optimize $A_\vartheta$ for the best performances (again measured with $\Gamma$), $\tau_\vartheta$ is defined as $\tau_\vartheta = \alpha/A_\vartheta$.

Figure 1A and B show kernels $\eta$ (step 1) and $\kappa$ (step 3) for a typical neuron. The double exponential shape of $\kappa$ is due to the coupling between somatic and dendritic compartments [15]. Figure 1C shows the optimal constant $\vartheta_{\text{cst}}$ plotted versus $f$. It is very well fitted by a simple linear function and allows to determine the parameters $\vartheta_0$ and $\alpha$ (steps 4 and 5).

## 2.3 Evaluation of performances

The performances of the model are evaluated with the coincidence factor $\Gamma$ [16]. It is defined by

$$\Gamma = \frac{N_{\text{coinc}} - \langle N_{\text{coinc}} \rangle}{\frac{1}{2}(N_{\text{data}} + N_{\text{SRM}})} \frac{1}{\mathcal{N}} \tag{5}$$

where $N_{\text{data}}$ is the number of spikes in the reference spike train, $N_{\text{SRM}}$ is the number of spikes in the predicted spike train $S_{\text{SRM}}$, $N_{\text{coinc}}$ is the number of coincidences with precision $\Delta$ between the two spike trains, and $\langle N_{\text{coinc}} \rangle = 2\nu\Delta N_{\text{data}}$ is the expected number of coincidences generated by a homogeneous Poisson process with the same rate $\nu$ as the spike train $S_{\text{SRM}}$. The factor $\mathcal{N} = 1 - 2\nu\Delta$ normalizes $\Gamma$ to a maximum value $\Gamma = 1$ which is reached if and only if the spike train of the SRM reproduces exactly that of the cell. A homogeneous Poisson process with the same number of spikes as the SRM would yield $\Gamma = 0$. We compute the coincidence factor $\Gamma$ by comparing the two complete spike trains as in [7]. Throughout the paper, we use $\Delta = 2\,\text{ms}$. Results do depend on $\Delta$ but the exact value of $\Delta$ is not critical as long as it is chosen in a reasonable range $1 \leq \Delta \leq 4\,\text{ms}$ [17]. The coincidence factor $\Gamma$ is similar to the "reliability" as defined in [6]. All measures of $\Gamma$

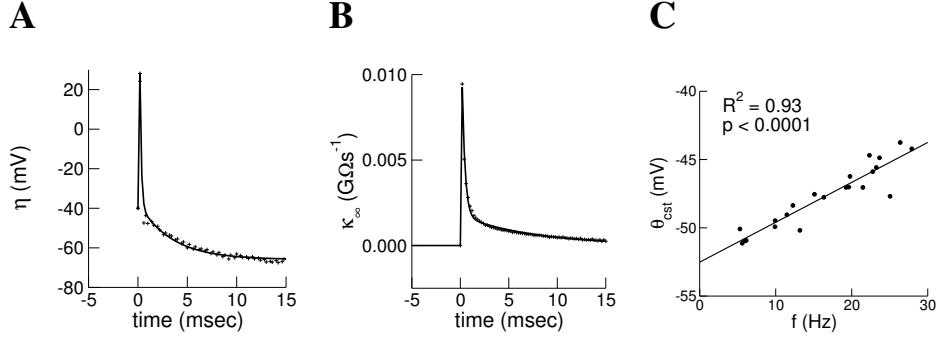

Figure 1: Kernels $\eta$ (**A**) and $\kappa$ (**B**) as extracted by the method exposed in this paper. Raw data (symbols) and fit by double exponential functions (solid line). **C.** The optimal constant threshold $\vartheta_{\text{cst}}$ is plotted versus the output frequency $f$ (symbols). It is very neatly fitted by a linear function (line).

reported in this paper are given for new stimuli, independent of those used for parameter optimization during the model estimation procedure.

## 3   Results

Figure 2 shows a direct comparison between predicted and recorded spike train for a typical neuron. Both spike trains are almost indistinguishable (A). Even when zooming on the subthreshold regime, differences are in the range of a few millivolts only (B). The spike dynamics is correctly predicted apart from a short period of time just after a spike is emitted (C). This is due to the fact that the kernel $\eta$ was extracted for a mean drive $\mu_I = 0$. Here, the mean is much larger than 0 and the neuron has already adapted to this new regime. It produces slightly different after-spike effects. This can be corrected easily in our framework by taking a time-dependent time constant in the kernel $\kappa$, i.e. $\kappa(s) \rightarrow \kappa(t - \hat{t}, s)$. This dependence is of importance to account for spike-to-spike interactions [18]. The mapping procedure discussed above allows, in principle, to compute $\kappa(t - \hat{t}, s)$ for any $t - \hat{t}$ (see [7] for further details). However, it requires longer recordings than the ones provided by our experiments and was dropped here.

Before moving to a quantitative estimate of the quality of the predictions of our model, we need to understand what kind of limits are imposed on predictions by the modelled neurons themselves. It is well known that pyramidal neurons of the cortex respond with very different reliability depending on the type of stimulation they receive [6]. Neurons tend to fire regularly but without conserving the exact timing of spikes in response to constant or quasi constant input current. On the other hand, they fire irregularly but reliably in terms of spike timing in response to fluctuating current. We do not expect our model to yield better predictions than the intrinsic reliability of the modelled neuron. To evaluate the intrinsic reliability of the pyramidal neurons, we repeated injection of the same OU process, i.e. injection of processes with the same seed, and computed $\Gamma$ between the repeated spike trains obtained in response to this procedure. Figure 3A shows a surface plot of the intrinsic reliability $\Gamma_{n \rightarrow n}$ of a typical neuron (the subscript $n \rightarrow n$ is written for *neuron to itself*). It is plotted versus the parameters of the stimulation, the current mean drive $\mu_I$ and its standard deviation $\sigma_I$. We find that the mean drive $\mu_I$ has almost no impact on $\Gamma_{n \rightarrow n}$ (measured cross-correlation coefficient $r = 0.04$ with a p-value $p = 0.81$). On the other hand, $\sigma_I$ has a strong impact on the reliability of the neuron ($r = 0.93$ with $p < 10^{-4}$). When $\sigma_I$ is large ($\sigma_I \gtrsim 300\,\text{pA}$), $\Gamma_{n \rightarrow n}$ reaches a plateau at about $0.84 \pm 0.05$ (mean $\pm$ s.d.).

**A**

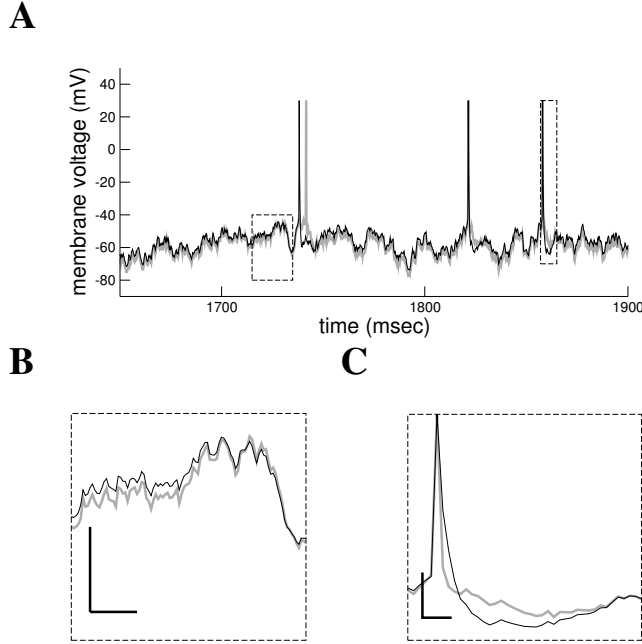

**B**                                    **C**

Figure 2: Performances of the SRM constructed by the method presented in this paper. **A.** The prediction of the model (black line) is compared to the spike train of the corresponding neuron (thick grey line). **B.** Zoom on the subthreshold regime. This panel corresponds to the first dotted zone in A (horizontal bar is 5 ms; vertical bar is 5 mV) **C.** Zoom on a correctly predicted spike. This panel corresponds to the second dotted zone in A (horizontal bar is 1 ms; vertical bar is 20 mV). The model slightly undershoots during about 4 ms after the spike (see text for further details).

When $\sigma_I$ decreases to $100 \leq \sigma_I \leq 300$ pA, $\Gamma_{n \to n}$ quickly drops to an intermediate value of $0.65 \pm 0.1$ and finally for $\sigma_I \leq 100$ pA drops down to $0.09 \pm 0.05$. These findings are stable across the different neurons that we recorded and repeat the findings of Mainen and Sejnowski [6].

In order to connect model predictions to these findings, we evaluate the $\Gamma$ coincidence factor between the predicted spike train and the recorded spike trains (this $\Gamma$ is labelled $m \to n$ for *model to neuron*). Figure 3B shows a plot of $\Gamma_{m \to n}$ versus $\Gamma_{n \to n}$. We find that the predictions of our minimal model are close to the natural upper bound set by the intrinsic reliability of the pyramidal neuron. On average, the minimal model achieves a quality $\Gamma_{m \to n}$ which is 65% ($\pm 3\%$ s.e.m.) of the upper bound, i.e. $\Gamma_{m \to n} = 0.65\,\Gamma_{n \to n}$. Furthermore, let us recall that due to the definition of the coincidence factor $\Gamma$, the threshold for statistical significance here is $\Gamma_{m \to n} = 0$. All the points are well above this value, hence highly significant. Finally, we compare the predictions of our minimal model in terms of two other indicators, the mean rate and the coefficient of variation of the interspike interval distribution ($C_v$). The mean rate is usually correctly predicted by our minimal model (see Figure 3C) in agreement with the findings of Rauch and colleagues [4]. The $C_v$ is predicted in the correct range as well but may vary due to missed or extra spikes added in the prediction (data not shown). It is also noteworthy that available spike trains are not very long (a few seconds) and the number of spikes is sometimes too low to yield a reliable estimate of the $C_v$.

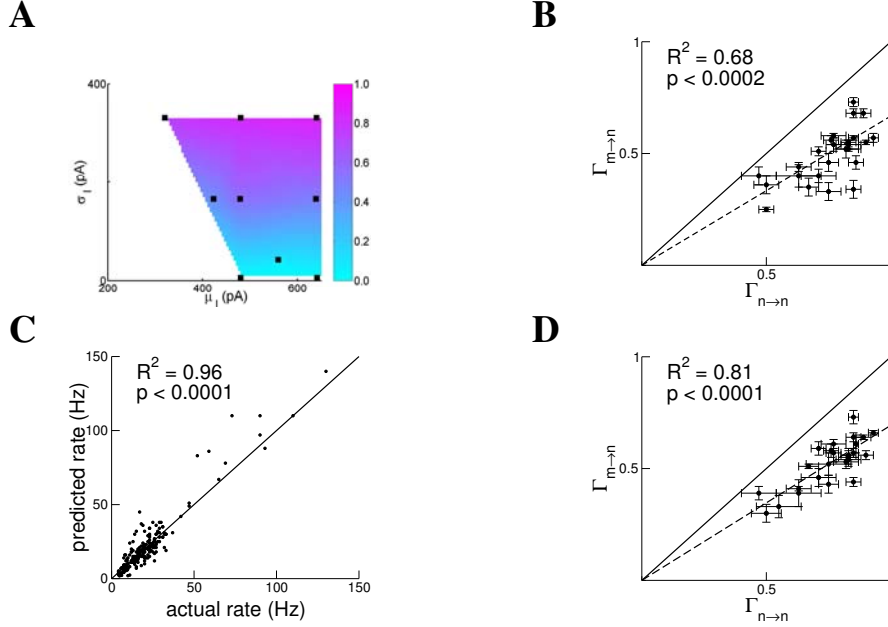

Figure 3: Quantitative performances of the model. **A.** Intrinsic reliability $\Gamma_{n \to n}$ of a typical pyramidal neuron in function of the mean drive $\mu_I$ and its standard deviation $\sigma_I$. **B.** Performances of the SRM in correct spike timing prediction $\Gamma_{m \to n}$ are plotted versus the cells intrinsic reliability $\Gamma_{n \to n}$ (symbols) for the very same stimulation parameters. The diagonal line (solid) denotes the "natural" upper bound limit imposed by the neurons intrinsic reliability. **C.** Predicted frequency versus actual frequency (symbols). **D.** Same as in A but in a model without adaptation where the threshold has been optimized separately for each set of stimulation parameters (see text for further details.)

Previous model studies had shown that a model with a threshold simpler than the one used here is able to reliably predict the spike train of more detailed neuron models [7,12]. Here, we used a threshold including an adaptation mechanism. Without adaptation, i.e. when the sum over all preceding spikes in Equation (3) is replaced by the contribution of the last emitted spike only, it is still possible to reach the same quality of predictions for each driving regime (Figure 3D) under the condition that the three threshold parameters ($A_\vartheta$, $\vartheta_0$ and $\tau_\vartheta$) are chosen differently for each set of input parameters $\mu_I$ and $\sigma_I$. In contrast to this, our I&F model with adaptation achieves the same level of predictive quality (Figure 3B) with one single set of threshold parameters. This illustrates the importance of adaptation to I&F models or SRM.

## 4   Discussion

Mapping real neurons to simplified neuronal models has benefited from many developments in recent years [4-5,7-8,11-13,19-22] and was applied to both in vitro [4,9,13,22] and in vivo recordings [5]. We have shown here that a simple estimation procedure allows to build an equivalent I&F-type model for a collection of cortical neurons. The model neuron is built sequentially from intracellular recordings. The resulting model is very efficient in the sense that it allows a quantitative and accurate prediction of the spike train of the real neuron. Most of the time, the predicted subthreshold membrane voltage differs from the

recorded one by a few millivolts only. The mean firing rate of the minimal model corresponds to that of the real neuron. The statistical structure of the spike train is approximately conserved since we observe that the coefficient of variation ($C_v$) of the interspike interval distribution is predicted in the correct range by our minimal model. But most important, our minimal model has the ability to predict spikes with the correct timing ($\pm 2\,\text{ms}$) and the level of prediction that is reached is close to the intrinsic reliability of the real neuron in terms of spike timing [6]. The adapting threshold has been found to play an important role. It allows the model to tune to variable input characteristics and to extend its predictions beyond the input regimes used for model evaluation.

This work suggests that L5 neocortical pyramidal neurons under random current injection behave very much like I&F neurons including a spike-frequency adaptation process. This is a result of importance. Indeed, the I&F-type models are extremely popular in large scale network studies. Our results can be viewed as a strong *a posteriori* justification to the use of this class of model neurons. They also indicate that the picture of a neuron combining a linear summation in the subthreshold regime with a threshold criterion for spike initiation is good enough to account for much of the behavior in an in vivo-like lab setting. This should however be moderated since several important aspects were neglected in this study.

First, we used random current injection rather than a more realistic random conductance protocol [23]. In a previous report [12], we had checked the consequences of random conductance injection with simulated data. We found that random conductance injection mainly changes the effective membrane time constant of the neuron and can be accounted for by making the time course of the optimal linear filter ($\kappa$ here) depend on the mean input to the neuron. The minimal model reached the same quality level of predictions when driven by random conductance injection [12] as the level it reaches when driven by random current injection [7]. Second, a largely fluctuating current generated by a random process can only be seen as a poor approximation to the input a neuron would receive in vivo. Our input has stationary statistics with a spectrum that is close to white (cut-off at $1\,\text{kHz}$), but a lower cut-off frequency could be used as well. Whether random input is a reasonable model of the input a neuron would receive in vivo is highly controversial [24-26], but from a purely practical point of view random stimulation provides at least a well-defined experimental paradigm for in vitro experiments that mimics some aspects of synaptic bombardment [27]. Third, all transient effects have been excluded since neuronal data is analyzed in the adapted state. Finally, our experimental paradigm used *somatic* current injection. Thus, all dendritic non-linearities, including backpropagating action potentials and dendritic spikes are excluded.

In summary, simple threshold models will never be able to account for all the variety of neuronal responses that can be probed in an artificial laboratory setting. For example, effects of delayed spike initiation cannot be reproduced by simple threshold models that combine linear subthreshold behavior with a strict threshold criterion (but could be reproduced by quadratic or exponential I&F models). For this reason, we are currently studying exponential I&F models with adaptation that allow us to relate our approach with other known models [21,28]. However, for random current injection that mimics synaptic bombardment, the picture of a neuron that combines linear summation with a threshold criterion is not too wrong. Moreover, in contrast to more complicated neuron models, the simple threshold model allows rapid parameter extraction from experimental traces; efficient numerical simulation; and rigorous mathematical analysis. Our results also suggest that, if any elaborated computation is taking place in single neurons, it is likely to happen at dendritic level rather than at somatic level. In absence of a clear understanding of dendritic computation, the I&F neuron with adaptation thus appears as a model that we consider "good enough".

## Acknowledgments

This work was supported by *Swiss National Science Foundation* grants number FN 200020-103530/1 to WG and number 3100-061335.00 to HRL.

## Footnotes

\* homepage: http://icwww.epfl.ch/~rjolivet

## References

[1] Feng J. *Neural Net.* **14**: 955–975, 2001.

[2] Maass W & Bishop C. *Pulsed Neural Networks*. MIT Press, Cambridge, 1998.

[3] Gerstner W & Kistler W. *Spiking neurons models: single neurons, populations, plasticity*. Cambridge Univ. Press, Cambridge, 2002.

[4] Rauch A, La Camera G, Lüscher H, Senn W & Fusi S. *J. Neurophysiol.* **90**: 1598–1612, 2003.

[5] Keat J, Reinagel P, Reid R & Meister M. *Neuron* **30**: 803-817, 2001.

[6] Mainen Z and Sejnowski T. *Science* **268**: 1503–1506, 1995.

[7] Jolivet R, Lewis TJ & Gerstner W. *J. Neurophysiol.* **92**: 959–976, 2004.

[8] Paninski L, Pillow J & Simoncelli E. *Neural Comp.* **16**: 2533-2561, 2004.

[9] Brillinger D & Segundo J. *Biol. Cyber.* **35**: 213-220, 1979.

[10] Benda J & Herz A. *Neural Comp.* **15**: 2523-2564, 2003.

[11] La Camera G, Rauch A, Lüscher H, Senn W & Fusi S. *Neural Comp.* **16**: 2101-2124, 2004.

[12] Jolivet R & Gerstner W. *J. Physiol.-Paris* **98**: 442-451, 2004.

[13] Jolivet R, Rauch A, Lüscher H & Gerstner W. *Accepted in J. Comp. Neuro.*

[14] Wiener N. *Nonlinear problems in random theory*. MIT Press, Cambridge, 1958.

[15] Roth A & Häusser M. *J. Physiol.* **535**: 445-472, 2001.

[16] Kistler W, Gerstner W & van Hemmen J. *Neural Comp.* **9**: 1015-1045, 1997.

[17] Jolivet R (2005). *Effective minimal threshold models of neuronal activity*. PhD thesis, EPFL, Lausanne.

[18] Arcas B & Fairhall A. *Neural Comp.* **15**: 1789-1807, 2003.

[19] Brillinger D. *Ann. Biomed. Engineer.* **16**: 3-16, 1988.

[20] Arcas B, Fairhall A & Bialek W. *Neural Comp.* **15**: 1715-1749, 2003.

[21] Izhikevich E. *IEEE Trans. Neural Net.* **14**: 1569-1572, 2003.

[22] Paninski L, Pillow J & Simoncelli E. *Neurocomp.* **65-66**: 379-385, 2005.

[23] Robinson H & Kawai N. *J. Neurosci. Meth.* **49**: 157-165, 1993.

[24] Arieli A, Sterkin A, Grinvald A & Aertsen A. *Science* **273**: 1868–1871, 1996.

[25] De Weese M & Zador A. *J. Neurosci.* **23**: 7940–7949, 2003.

[26] Stevens C & Zador A. In *Proc. of the 5th Joint Symp. on Neural Comp.*, Inst. for Neural Comp., La Jolla, 1998.

[27] Destexhe A, Rudolph M & Paré D. *Nat. Rev. Neurosci.* **4**: 739-751, 2003.

[28] Fourcaud-Trocmé N, Hansel D, van Vreeswijk C & Brunel N. *J. Neurosci.* **23**: 11628-11640, 2003.
